# Identifying Fault-Prone Software Modules Using Feed-Forward Networks: A Case Study

**N. Karunanithi**
Room 2E-378, Bellcore
435 South Street
Morristown, NJ 07960
E-mail: *karun@faline.bellcore.com*

## Abstract

Functional complexity of a software module can be measured in terms of *static complexity metrics* of the program text. Classifying software modules, based on their static complexity measures, into different fault-prone categories is a difficult problem in software engineering. This research investigates the applicability of neural network classifiers for identifying fault-prone software modules using a data set from a commercial software system. A preliminary empirical comparison is performed between a minimum distance based Gaussian classifier, a perceptron classifier and a multilayer layer feed-forward network classifier constructed using a modified Cascade-Correlation algorithm. The modified version of the Cascade-Correlation algorithm constrains the growth of the network size by incorporating a cross-validation check during the output layer training phase. Our preliminary results suggest that a multilayer feed-forward network can be used as a tool for identifying fault-prone software modules early during the development cycle. Other issues such as representation of software metrics and selection of a proper training samples are also discussed.

# 1  Problem Statement

Developing reliable software at a low cost is an important issue in the area of software engineering (Karunanithi, Whitley and Malaiya, 1992). Both the reliability of a software system and the development cost can be reduced by identifying troublesome software modules early during the development cycle. Many measurable program attributes have been identified and studied to characterize the intrinsic complexity and the fault proneness of software systems. The intuition behind software complexity metrics is that complex program modules tend to be more error prone than simple modules. By controlling the complexity of software modules during development, one can produce software systems that are easy to maintain and enhance (because simple program modules are easy to understand). Static complexity metrics are measured from the passive program texts early during the development cycle and can be used as a valuable feedback for allocating resources in future development efforts (future releases or new projects).

Two approachs can be applied to relate static complexity measures with faults found or program changes made during testing. In the *estimative approach* regressions models are used to predict the actual number of faults that will be disclosed during testing (Lipow, 1982; Gaffney, 1984; Shen *et al.*, 1985; Crawford *et al.*, 1985; Munson and Khoshgoftaar, 1992). Regression models assume that the metrics that constitute independent variables are independent and normally distributed. However, most practical measures often violate the normality assumptions and exhibit high correlation with other metrics (i.e., multicollinearity). The resulting fit of the regression models often tend to produce inconsistent predictions.

Under the *classification approach* software modules are categorized into two or more fault-prone classes (Rodriguez and Tsai, 1987; Munson and Khoshgoftaar, 1992; Karunanithi, 1993; Khoshgoftaar *et al.*, 1993). A special case of the classification approach is to classify software modules into either low-fault (non-complex) or high-fault (complex) categories. The main rationale behind this approach is that the software managers are often interested in getting some approximate feedback from this type of models rather than accurate predictions of the number of faults that will be disclosed. Existing two-class categorization models are based on linear discriminant principle (Rodriguez and Tsai, 1987; Munson and Khoshgoftaar, 1992). Linear discriminant models assume that the metrics are orthogonal and that they follow a normal distribution. To reduce multicollinearity, researchers often use principle component analysis or some other dimensionality reduction techniques. However, the reduced metrics may not explain all the variability if the original metrics have nonlinear relationship.

In this paper, the applicability of neural network classifiers for identifying fault proneness of software modules is examined. The motivation behind this research is to evaluate whether classifiers can be developed without usual assumptions about the input metrics. In order to study the usefulness of neural network classifiers, a preliminary comparison is made between a simple minimum distance based Gaussian classifier, a single layer perceptron and a multilayer feed-forward network developed using a modified version of Fahlman's Cascade Correlation algorithm (Fahlman and Lebiere, 1990). The modified algorithm incorporates a cross-validation for constraining the growth of the size of the network. In this investigation, other issues

such as selection of proper training samples and representation of metrics are also considered.

# 2    Data Set Used

The metrics data used in this study were obtained from a research conducted by Lind and Vairavan (Lind and Vairavan, 1989) for a Medical Imaging System software. The complete system consisted of approximately 4500 modules amounting to about 400,000 lines of code written in Pascal, FORTRAN, PL/M and assembly level. From this set, a random sample of 390 high level language routines was selected for the analysis. For each module in the sample, program changes were recorded as an indication of software fault. The number of changes in the program modules varied from zero to 98. In addition to changes, 11 software complexity metrics were extracted from each module. These metrics range from total lines of code to Belady's bandwidth metric. (Readers curious about these metrics may refer to Table I of Lind and Vairavan, 1989.) For the purpose of our classification study, these metrics represent 11 input (both real and integer) variables of the classifier.

A software module is considered as a low fault-prone module (Category I) if there are 0 or 1 changes and as a high fault-prone module (Category II) if there are 10 or more changes. The remaining modules are considered as medium fault category. For the purpose of this study we consider only the low and high fault-prone modules. Our extreme categorization and deliberate discarding of program modules is similar to the approach used in other studies (Rodriguez and Tsai, 1987; Munson and Khoshgoftaar, 1992). After discarding medium fault-prone modules, there are 203 modules left in the data set. Of 203 modules, 114 modules belong to the low fault-prone category while the remaining 89 modules belong to the high fault-prone category. The output layer of the neural nets had two units corresponding to two fault categories.

# 3    Training Data Selection

We had two objectives in selecting training data: 1) to evaluate how well a neural network classifier will perform across different sized training sets and 2) to select the training data as much unbiased as possible. The first objective was motivated by the need to evaluate whether a neural network classifier can be used early in the software development cycle. Thus the classification experiments were conducted using training samples of size $S = \frac{1}{4}, \frac{1}{3}, \frac{1}{2}, \frac{2}{3}, \frac{3}{4}, \frac{9}{10}$ fraction of 203 samples belonging to Categories I and II. The remaining (1-$S$) fraction of the samples were used for testing the classifiers. In order to avoid bias in the training data, we randomly selected 10 different training samples for each fraction $S$. This resulted in 6 X 10 (=60) different training and test sets.

## 4   Classifiers Compared

### 4.1   A Minimum Distance Classifier

In order to compare neural network classifiers and linear discriminant classifiers we implemented a simple minimum distance based two-class Gaussian classifier of the form (Nilsson, 1990):

$$|X - C_i| = ((X - C_i)(X - C_i)^t)^{1/2}$$

where $C_i$, $i = 1, 2$ represent the prototype points for the Categories I and II, $X$ is a 11 dimensional metrics vector, and $t$ is the transpose operator. The prototype points $C_1$ and $C_2$ are calculated from the training set based on the normality assumption. In this approach a given arbitrary input vector $X$ is placed in Category I if $|X - C_1| < |X - C_2|$ and in Category II otherwise.

All raw component metrics had distributions that are asymmetric with a positive skew (i.e., long tail to the right) and they had different numerical ranges. Note that asymmetric distributions do not conform to the normality assumption of a typical Gaussian classifier. First, to remove the extreme asymmetry of the original distribution of the individual metric we transformed each metric using a natural logarithmic base. Second, to mask the influence of individual component metric on the distance score, we divided each metric by its standard deviation of the training set. These transformations considerably improved the performance of the Gaussian classifier. To be consistent in our comparison we used the log transformed inputs for other classifiers also.

### 4.2   A Perceptron Classifier

A perceptron with a hard-limiting threshold can be considered as a realization of a non-parametric linear discriminant classifier. If we use a sigmoidal unit, then the continuous valued output of the perceptron can be interpreted as a likelihood or probability with which inputs are assigned to different classes. In our experiment we implemented a perceptron with two sigmoidal units (outputs 1 and 2) corresponding to two categories. A given arbitrary vector **X** is assigned to Category I if the value of the output unit 1 is greater than the output of the unit 2 and to Category II otherwise. The weights of the network are determined iteratively using least square error minimization procedure. In almost all our experiments, the perceptron learned about 75 to 80 percentages of the training set. This implies that the rest of the training samples are not linearly separable.

### 4.3   A Multilayer Network Classifier

To evaluate whether a multilayer network can perform better than the other two classifiers, we repeated the same set of experiments using feed-forward networks constructed by Fahlman's Cascade-Correlation algorithm. The Cascade-Correlation algorithm is a constructive training algorithm which constructs a suitable network architecture by adding one hidden (layer) unit at a time. (Refer to Fahlman and Lebiere, 1990 for more details on the Cascade-Correlation algorithm.) Our initial results suggested that the multilayer layer networks constructed by the Cascade-Correlation algorithm are not capable of producing a better classification accuracy

than the other two classifiers. An analysis of the network suggested that the resulting networks had too many free variables (i.e., due to too many hidden units). A further analysis of the rate of decrease of the residual error versus the number of hidden units added to the networks revealed that the Cascade-Correlation algorithm is capable of adding more hidden units to learn individual training patterns at the later stages of the training phase than in the earlier stages. This happens if the training set contains patterns that are interspersed across different decision regions or what might be called "border patterns" (Ahmed, S. and Tesauro, 1989). In an effort to constrain the growth of the size of the network, we modified the Cascade-Correlation algorithm to incorporate a cross-validation check during the output layer training phase. For each training set of size **S**, one third was used for cross-validation and the remaining two third was used to train the network. The network construction was stopped as soon as the residual error of the cross-validation set stopped decreasing from the residual error at the end of the previous output layer training phase. The resulting network learned about 95% of the training patterns. However, the cross-validated construction considerably improved the classification performance of the networks on the test set. Table 1 presented in the next section provides a comparison between the networks developed with and without cross-validation.

| Training Set Size S in% | Hidden Unit Statistics | | Error Statistics | | | |
|---|---|---|---|---|---|---|
| | Mean | Std | Type I Error | | Type II Error | |
| | | | Mean | Std | Mean | Std |
| Without Cross-Validation | | | | | | |
| 25 | 5.1 | 1.5 | 24.64 | 7.2 | 16.38 | 6.4 |
| 33 | 6.2 | 1.8 | 20.24 | 8.4 | 17.27 | 5.5 |
| 50 | 7.4 | 1.8 | 18.30 | 7.4 | 18.65 | 6.4 |
| 67 | 9.7 | 1.7 | 15.78 | 6.5 | 18.05 | 7.1 |
| 75 | 10.4 | 1.8 | 14.54 | 7.6 | 16.85 | 7.3 |
| 90 | 11.2 | 1.6 | 10.33 | 7.2 | 17.73 | 8.3 |
| With Cross-Validation | | | | | | |
| 25 | 1.9 | 1.3 | 20.19 | 5.4 | 12.11 | 4.7 |
| 33 | 2.2 | 1.0 | 18.24 | 5.5 | 12.40 | 4.1 |
| 50 | 2.0 | 0.9 | 17.41 | 5.6 | 15.04 | 5.2 |
| 67 | 2.7 | 1.1 | 14.32 | 5.8 | 14.08 | 5.5 |
| 75 | 2.7 | 1.3 | 13.27 | 7.0 | 13.84 | 5.4 |
| 90 | 2.9 | 1.2 | 9.77 | 9.4 | 15.47 | 5.1 |

Table 1: A Comparison of Nets With and Without Cross-Validation.

## 5  Results

In this section we present some preliminary results from our classification experiments. First, we provide a comparison between the multilayer networks developed with and without cross-validation. Next, we compare different classifiers in terms of their classification accuracy. Since a neural network's performance can be affected by the weight vector used to initialize the network, we repeated the training experiment 25 times with different initial weight vectors for each training set. This

resulted in a total of 250 training trials for each value of **S**. The results reported here for the neural network classifiers represent a summary statistics for 250 experiments.

The performance of the classifiers are reported in terms of classification errors. There are two type of classification errors that a classifier can make: a Type I error occurs when the classifier identifies a low fault-prone (Category I) module as a high fault-prone (Category II) module; a Type II error is produced when a high fault-prone module is identified as a low fault-prone module. From a software manager's point of view, these classification errors will have different implications. Type I misclassification will result in waste of test resources (because modules that are less fault-prone may be tested longer than what is normally required). On the other hand, Type II misclassification will result in releasing products that are of inferior quality. From reliability point of view, a Type II error is a serious error than a Type I error.

|  | No. of Patterns | | Error Statistics | | | | | |
|---|---|---|---|---|---|---|---|---|
| **S** % | Training Set | Test Set | Gaussian | | Perceptron | | Multilayer Nets | |
| | | | Mean | Std | Mean | Std | Mean | Std |
| Type I Error Statistics | | | | | | | | |
| 25 | 50 | 86 | 13.16 | 4.7 | 16.17 | 5.5 | 20.19 | 5.4 |
| 33 | 66 | 77 | 11.44 | 4.0 | 11.74 | 3.9 | 18.24 | 5.5 |
| 50 | 101 | 57 | 12.45 | 3.2 | 11.58 | 3.2 | 17.41 | 5.6 |
| 67 | 136 | 37 | 9.46 | 4.1 | 10.14 | 3.9 | 14.32 | 5.8 |
| 75 | 152 | 28 | 8.57 | 5.4 | 9.15 | 5.8 | 13.27 | 7.0 |
| 90 | 182 | 12 | 14.17 | 7.9 | 4.03 | 4.3 | 9.77 | 9.4 |
| Type II Error Statistics | | | | | | | | |
| 25 | 50 | 67 | 15.61 | 4.2 | 15.98 | 7.8 | 12.11 | 4.7 |
| 33 | 66 | 60 | 15.46 | 4.6 | 15.78 | 6.6 | 12.40 | 4.1 |
| 50 | 101 | 45 | 16.01 | 5.1 | 16.97 | 6.8 | 15.04 | 5.2 |
| 67 | 136 | 30 | 16.00 | 5.4 | 16.11 | 7.6 | 14.08 | 5.5 |
| 75 | 152 | 23 | 17.39 | 5.8 | 18.39 | 6.3 | 13.84 | 5.4 |
| 90 | 182 | 9 | 21.11 | 6.3 | 19.11 | 5.6 | 15.47 | 5.1 |

Table 2: A Summary of Type I and Type II Error Statistics.

Table 1 compares the complexity and the performance of the multilayer networks developed with and without cross-validation. Columns 2 through 7 represent the size and the performance of the networks developed by the Cascade-Correlation without cross-validation. The remaining six columns correspond to the networks constructed with cross-validation. Hidden unit statistics for the networks suggest that the growth of the network can be constrained by adding a cross-validation during the output layer training. The corresponding error statistics for both the Type I and Type II errors suggest that an improvement classification accuracy can be achieved by cross-validating the size of the networks.

Table 2 illustrates the preliminary results for different classifiers. The first two columns in Table 2 represent the size of the training set in terms of **S** as a percentage of all patterns and the number of patterns respectively. The third column represents the number of test patterns in Categories I (1st half) and the II (2nd half). The remaining six columns represent the error statistics for the three classifiers in

terms of percentage mean errors and standard deviations. The percentages errors were obtained by dividing the number of misclassifications by the total number of test patterns in that Category. The Type I error statistics in the first half of the table suggest that the Gaussian and the Perceptron classifiers may be better than multilayer networks at early stages of the software development cycle. However, the difference in performance of the Gaussian classifier is not consistent across all values of **S**. The neural network classifiers seem to improve their performance with an increase in the size of the training set. Among neural networks, the perceptron classifier seems to perform classification than a multilayer net. However, the Type II error statistics in the second half of the table suggest that a multilayer network classifier may provide a better classification of Category II modules than the other two classifiers. *This is an important results from the reliability perspective.*

## 6   Conclusion and Work in Progress

We demonstrated the applicability of neural network classifiers for identifying fault-prone software modules. We compared the classification efficacy of three different pattern classifiers using a data set from a commercial software system. Our preliminary empirical results are encouraging in that there is a role for multilayer feed-forward networks either during the software development cycle of a subsequent release or for a similar product.

The cross-validation implemented in our study is a simple heuristics for constraining the size of the networks constructed by the Cascade-Correlation algorithm. Though this improved the performance of the resulting networks, it should be cautioned that cross-validation may be needed only if the training patterns exhibit certain characteristics. In other circumstances, the networks may have to be constructed using the entire training set. At this stage we have not performed complete analysis on what characteristics of the training samples would require cross-validation for constraining the network growth. Also we have not used other sophisticated structure reduction techniques. We are currently exploring different loss functions and structure reduction techniques.

The Cascade-Correlation algorithm always constructs a deep network. Each additional hidden unit develops an internal representation that is a higher order sigmoidal computation than those of previously added hidden units. Such a complex internal representation may not be appropriate in a classification application such as the one studied here. We are currently exploring alternatives to construct shallow networks within the Cascade-Correlation frame work.

At this stage, we have not performed any analysis on how the internal representations of a multilayer network correlate with the input metrics. This is currently being studied.

### References

Ahmed, S. and G. Tesauro (1989). "Scaling and Generalization in Neural Networks: A Case Study", *Advances in Neural Information Processing Systems 1*, pp 160-168, D. Touretzky, ed. Morgan Kaufmann.

Crawford, S. G., McIntosh, A. A. and D. Pregibon (1985). "An Analysis of Static Metrics and Faults in C Software", *The Journal of Systems and Software*, Vol. 5, pp. 37-48.

Fahlman, S. E. and C. Lebiere (1990). "The Cascaded-Correlation Learning Architecture," *Advances in Neural Information Processing Systems 2*, pp 524-532, D. Touretzky, ed. Morgan Kaufmann.

Gaffney Jr., J. E. (1984). "Estimating the Number of Faults in Code", *IEEE Trans. on Software Eng.*, Vol. SE-10, No. 4, pp. 459-464.

Karunanithi, N, Whitley, D. and Y. K. Malaiya (1992). "Prediction of Software Reliability Using Connectionist Models", *IEEE Trans. on Software Eng.*, Vol. 18, No. 7, pp. 563-574.

Karunanithi, N. (1993). "Identifying Fault-Prone Software Modules Using Connectionist Networks", *Proc. of the 1st Int'l Workshop on Applications of Neural Networks to Telecommunications*, (IWANNT'93), pp. 266-272, J. Alspector *et al.*, ed., Lawrence Erlbaum, Publisher.

Khoshgoftaar, T. M., Lanning, D. L. and A. S. Pandya (1993). "A Neural Network Modeling Methodology for the Detection of High-Risk Programs", *Proc. of the 4th Int'l Symp. on Software Reliability Eng.* pp. 302-309.

Lind, R. K. and K. Vairavan (1989). "An Experimental Investigation of Software Metrics and Their Relationship to Software Development Effort", *IEEE Trans. on Software Eng.*, Vol. 15, No. 5, pp. 649-653.

Lipow, M. (1982). "Number of Faults Per Line of Code", *IEEE Trans. on Software Eng.*, Vol. SE-8, No. 4, pp. 437-439.

Munson, J. C. and T. M. Khoshgoftaar (1992). "The Detection of Fault-Prone Programs", *IEEE Trans. on Software Eng.*, Vol. 18, No. 5, pp. 423-433.

Nilsson, J. Nils (1990). *The Mathematical Foundations of Learning Machines*, Morgan Kaufmann, Chapters 2 and 3.

Rodriguez, V. and W. T. Tsai (1987). "A Tool for Discriminant Analysis and Classification of Software Metrics", *Information and Software Technology*, Vol. 29, No. 3, pp. 137-149.

Shen, V. Y., Yu, T., Thebaut, S. M. and T. R. Paulsen (1985). "Identifying Error-Prone Software: An Empirical Study", *IEEE Trans. on Software Eng.*, Vol. SE-11, No. 4, pp. 317-323.